# Neural Basis of Object-Centered Representations

**Sophie Deneve and Alexandre Pouget**
Georgetown Institute for Computational and Cognitive Sciences
Georgetown University
Washington, DC 20007-2197
sophie, alex@giccs.georgetown.edu

## Abstract

We present a neural model that can perform eye movements to a particular side of an object regardless of the position and orientation of the object in space, a generalization of a task which has been recently used by Olson and Gettner [4] to investigate the neural structure of object-centered representations. Our model uses an intermediate representation in which units have oculocentric receptive fields– just like collicular neurons— whose gain is modulated by the side of the object to which the movement is directed, as well as the orientation of the object. We show that these gain modulations are consistent with Olson and Gettner's single cell recordings in the supplementary eye field. This demonstrates that it is possible to perform an object-centered task without a representation involving an object-centered map, viz., without neurons whose receptive fields are defined in object-centered coordinates. We also show that the same approach can account for object-centered neglect, a situation in which patients with a right parietal lesion neglect the left side of objects regardless of the orientation of the objects.

Several authors have argued that tasks such as object recognition [3] and manipulation [4] are easier to perform if the object is represented in object-centered coordinates, a representation in which the subparts of the object are encoded with respect to a frame of reference centered on the object. Compelling evidence for the existence of such representations in the cortex comes from experiments on hemineglect— a neurological syndrome resulting from unilateral lesions of the parietal cortex such that a right lesion, for example, leads patients to ignore stimuli located on the left side of their egocentric space. Recently, Driver et al. (1994) showed that the deficit can also be object-centered. Hence, hemineglect patients can detect a gap in the upper edge of a triangle when this gap is associated with the right side of the object

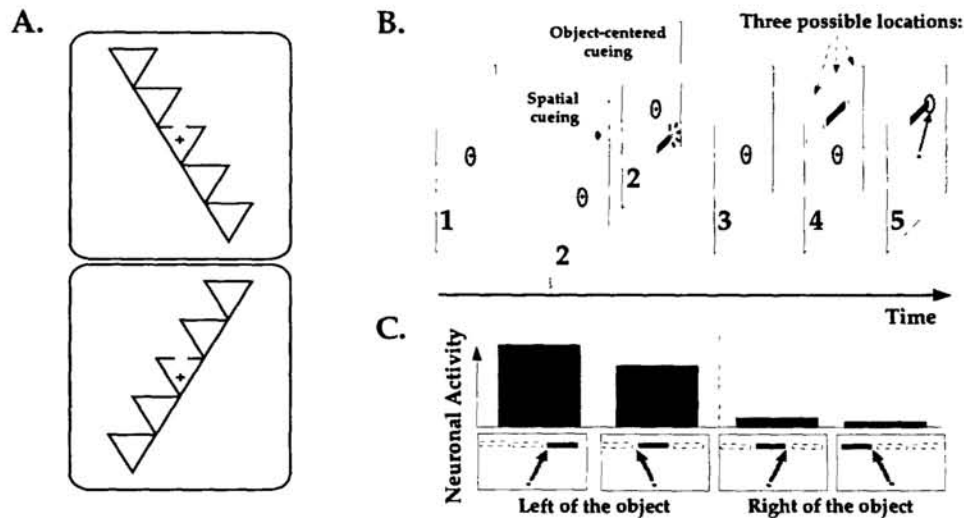

Figure 1: A- Driver et al. (1994) experiment demonstrating object-centered neglect. Subjects were asked to detect a gap in the upper part of the middle triangle, while fixating at the cross, when the overall figure is tilted clockwise (top) or counterclockwise (bottom). Patients perform worse for the clockwise condition, when the gap is perceived to be on the left of the overall figure. B- Sequence of screens presented on each trial in Olson and Gettner experiment (1995). 1- Fixation, 2- apparition of a cue indicating where the saccade should go, either in object-centered coordinates (object-centered cueing), or in screen coordinates (spatial cueing), 3- delay period, 4- apparition of the bar in one of three possible locations (dotted lines), and 5- saccade to the cued location. C- Schematic response of an SEF neuron for 4 different conditions. Adapted from [4].

but not when it belongs to the left side (figure 1-A).

What could be the neural basis of these object-centered representations? The simplest scheme would involve neurons with receptive fields defined in object-centered coordinates, i.e., the cells respond to a particular side of an object regardless of the position and orientation of the object. A recent experiment by Olson and Gettner (1995) supports this possibility. They recorded the activity of neurons in the supplementary eye field (SEF) while the monkey was performing object-directed saccades. The task consisted of making a saccade to the right or left side of a bar, independently of the position of the bar on the screen and according to the instruction provided by a visual cue. For instance, the cue corresponding to the instruction 'Go to the right side of the bar' was provided by highlighting the right side of a small bar briefly flashed at the beginning of the trial (step 2 in figure 1-B).

By changing the position of the object on the screen, it is possible to compare the activity of a neuron for movements involving different saccade directions but directed to the same side of the object, and vice-versa, for movements involving the same saccade direction but directed to opposite sides of the object. Olson and Gettner found that many neurons responded more prior to saccades directed to a particular side of the bar, independently of the direction of the saccades. For example, some neurons responded more for an upward right saccade directed to the *left* side of the bar but not at all for the same upward right saccade directed to the *right* side of the bar (column 1 and 3, figure 1-C). This would suggest that these neurons have bar-centered receptive[1] fields, i.e., their receptive fields are centered

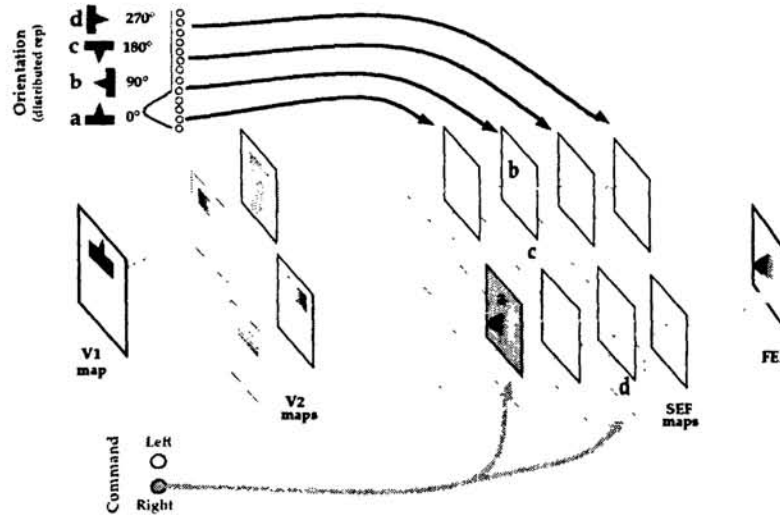

Figure 2: Schematic structure of the network with activity patterns in response to the horizontal bar shown in the V1 map and the command 'Go to the right'. Only one SEF map is active in this case, the one selective to the right edge of the bar (where right is defined in retinal coordinates), object orientation of 0° and the command 'Go to the right'. The letter a, b, c and d indicate which map would be active for the same command but for various orientations of the object, respectively, 0°, 90°, 180°, 270°. The dotted lines on the maps indicate the outline of the bar. Only a few representative connections are shown.

on the bar and not on the retina. This would correspond to what we will call an *explicit* object-centered representation.

We argue in this paper that these data are compatible with a different type of representation which is more suitable for the task performed by the monkey. We describe a neural network which can perform a saccade to the right, or left, boundary of an object, regardless of its orientation, position or size— a generalization of the task used by Olson and Gettner. This network uses units with receptive fields defined in oculocentric coordinates, i.e., they are selective for the direction and amplitude of saccades with respect to the fixation point, just like collicular neurons. These tuning curves, however, are also modulated by two types of signals, the orientation of the object, and the command indicating the side of the object to which the saccade should be directed. We show that these response properties are compatible with the Olson and Gettner data and provide predictions for future experiments. We also show that a simulated lesion leads to object-centered neglect as observed by Driver et al. (1994).

## 1   Network Architecture

The network performs a mapping from the image of the bar and the command (indicating the side of the object to which the saccade must be directed) to the appropriate motor command in oculocentric coordinates (the kind of command observed in the frontal eye field, FEF). We use a bar whose left and right sides are defined with respect to a a triangle appearing on the top of the bar (see figure 2).

The network is composed of four parts. The first two parts of the network models the

---

field.

lower areas in visual cortex, where visual features are segmented within retinotopic maps. In the first layer, the image is projected on a very simple V1-like map (10 by 10 neurons with activity equal to one if a visual feature appears within their receptive field, and zero otherwise). The second part on the network contains 4 different V2 retinotopic maps, responding respectively to the right, left, top and bottom boundary of the bar.

This model of V2 is intended to reproduce the response properties of a subset of cells recently discovered by Zhou et al. (1996). These cells respond to oriented edges, like V1 cells, but when the edge belongs to a closed figure, they also show a selectivity for the side on which the figure appears with respect to the edge. For example, a cell might respond to a vertical edge only if this edge is on the right side of the figure but not on the left (where right and left are defined with respect to the viewer, not the object itself). This was observed for any orientation of the edge, but we limit ourselves in this model to horizontal and vertical ones.

The third part of the network models the SEF and is divided into 4 groups of 4 maps, each group receiving connections from the corresponding map in V2 (figure 2). Within each group of maps, visual activity is modulated by signals related to the orientation of the object (assumed to be computed in temporal cortex) such that each of the 4 maps respond best for one particular orientation (respectively $0°$, $90°$, $180°$ and $270°$). For example, a neuron in the second map of the top group responds maximally if: 1- there is an edge in its receptive field and the figure is below, and 2- the object has an orientation of $90°$ counterclockwise. Note that this situation arises *only* if the left side of the object appears in the cell's receptive field; it will never occur for the right side. However, the cell is only partially selective to the left side of the object, e.g., it does not respond when the left side is in the retinal receptive field and the orientation of the object is $270°$ counterclockwise.

These collection of responses can be used to generate an object-centered saccade by selecting the maps which are partially selective for the side of the object specified by the command. This is implemented in our network by modulating the SEF maps by signals related to the command. For example, the unit encoding 'go to the right' send a positive weight to any map compatible with the right side of the object while inhibiting the other maps (figure 2).

Therefore, the activity, $B_{ij}^k$, of a neuron at position $ij$ on the map $k$ in the SEF is the product of three functions:

$$B_{ij}^k = V_{ij} f_k(\theta) g_k(C)$$

where $V_{ij}$ is the visual receptive field from the V2 map, $f_k(\theta)$ is a gaussian function of orientation centered on the cell preferred orientation, $\theta_k$, and $g_k(C)$ is the modulation by the command unit.

Olson and Gettner also used a condition with spatial cueing, viz., the command was provided by a spatial cue indicating where the saccade should go, as opposed to an object-centered instruction (see figure 1-B). We modeled this condition by simply multiplying the activity of neurons coding for this location in all the SEF maps by a fixed constant (10 in the simulations presented here). We also assume that there is no modulation by orientation of the object since this information is irrelevant in this experimental condition.

Finally, the fourth part of the network consists of an oculocentric map similar to the one found in the frontal eye field (FEF) or superior colliculus (SC) in which the command for the saccade is generated in oculocentric coordinates. The activity in the output map, $\{O_{ij}\}$, is obtained by simply summing the activities of all the

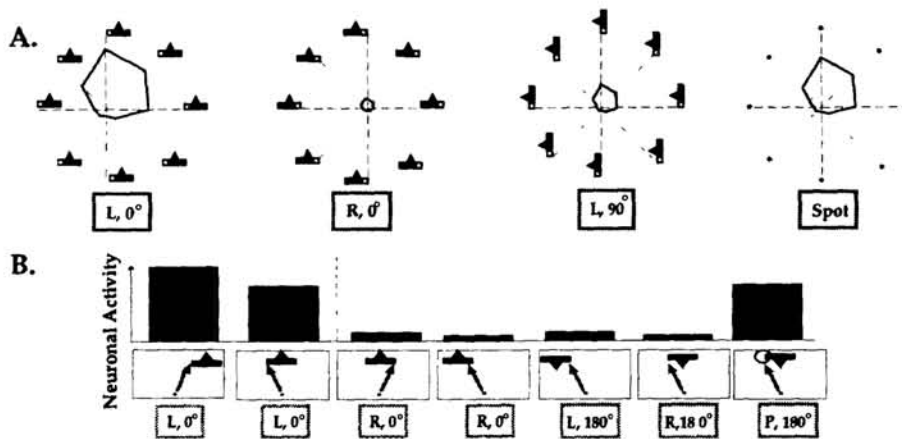

Figure 3: A- Polar plots showing the selectivity for saccade direction of a representative SEF units. The first three plots are for various combinations of command (L: left, R: right, P: spatial cueing) and object orientation. The left plot corresponds to saccades to a single dot. B- Data for the same unit but for a subset of the conditions. The first four columns can be directly compared to the experimental data plotted in figure 1-C from Olson and Gettner. The 5th and 6th columns show responses when the object is inverted. The seventh columns corresponds to spatial cueing.

SEF maps. The result is typically a broad two dimensional bell-shaped pattern of activity from which one can read out the horizontal and vertical components, $x_s$ and $y_s$, of the intended saccade by applying a center-of-mass operator.

$$x_s = \frac{\sum_i x_i O_{ij}}{\sum_{ij} O_{ij}}, \quad y_s = \frac{\sum_j y_j O_{ij}}{\sum_{ij} O_{ij}} \qquad (1)$$

## 2   Results

This network is able to generate a saccade to the right or to left of a bar, whatever its position, size and orientation. This architecture is basically like a radial basis function network, i.e., a look-up table with broad tuning curves allowing for interpolation. Consequently, one or two of the SEF maps light up at the appropriate location for any combination of the command and, position and orientation of the bar.

Neurons in the SEF maps have the following property: 1- they have an invariant tuning curve for the direction (and amplitude) of saccadic eye movements in oculocentric coordinates, just like neurons in the FEF or the SC, and 2- the gain of this tuning curve is modulated by the orientation of the object as well as the command. Figure 3-A shows how the tuning curve for saccade direction of one particular unit varies as a function of these two variables. In this particular case, the cell responds best to a right-upward saccade directed to the left side of the object when the object is horizontal. Therefore, the SEF units in our model do not have an invariant receptive field in object-centered coordinates but, nevertheless, the gain modulation is sufficient to perform the object-centered saccade. We predict that similar response properties should be found in the SEF, and perhaps parietal cortex, of a monkey trained on a task analogous to the one described here.

Since Olson and Gettner (1995) tested only three positions of the bar and held its

orientation constant, we cannot determine from their data whether SEF neurons are gain modulated in the way we just described. However, they found that all the SEF neurons had oculocentric receptive fields when tested on saccades to a single dot (personal communication), an observation which is consistent with our hypothesis (see fourth plot in figure 3-A) while being difficult to reconcile with an explicit object-centered representation. Second, if we sample our data for the conditions used by Olson and Gettner, we find that our units behave like real SEF cells. The first four columns of figure 3-B shows a unit with the same response properties as the cell represented in figure 1-C. Figure 3-B also shows the response of the same unit when the object is upside down and when we use a spatial cue. Note that this unit responds to the left side of the bar when the object is upright (1st column), but not when it is rotated by 180° (5th column), unless we used a spatial cue (7th column). The absence of response for the inverted object is due to the selectivity of the cell to orientation. The cell nevertheless responds in the spatial cueing conditions because we have assumed that orientation does not modulate the activity of the units in this case, since it is irrelevant to the task.

Therefore, the gain modulation observed in our units is consistent with available experimental data and makes predictions for future experiments.

## 3  Simulation of Neglect

Our representational scheme can account for neglect if the parietal cortex contains gain modulated cells like the ones we have described and if each cortical hemisphere contains more units selective for the contralateral side of space. This is known to be the case for the retinal input; hence most cells in the left hemisphere have their receptive field on the right hemiretina. We propose that the left hemisphere also over-represents the right side of objects and vice-versa (where right is defined in object-centered coordinates).

Recall that the SEF maps in our model are partially selective for the side of objects. A hemispheric preference for the contralateral side of objects could therefore be achieved by having all the maps responding to the left side of objects in the right hemisphere. Clearly, in this case, a right lesion would lead to left object-centered neglect; our network would no longer be able to perform a saccade to the left side of an object.

If we add the retinal gradient and make the previous gradient not quite as binary, then we predict that a left lesion leads to a syndrome in which the network has difficulty with saccades to the left side of an object but more so if the object is shown in the left hemiretina. Preliminary data from Olson and Gettner (personal communication) are compatible with this prediction.

The same model can also account for Driver et al. (1994) experiment depicted in figure 1-A. If the hemispheric gradients are as we propose, a right parietal lesion would lead to a situation in which the overall activity associated with the gap, i.e., the summed activity of all the neurons responding to this retinal location, is greater when the object is rotated counterclockwise— the condition in which the gap is perceived as belonging to the right side of the object— than in the clockwise condition. This activity difference, which can be thought as being a difference in the saliency of the upper edge of the triangle, may be sufficient to account for patients' performance.

Note that object-centered neglect should be observed only if the orientation of the object is taken into consideration by the SEF units. If the experimental conditions

are such that the orientation of the object can be ignored by the subject —a situation similar to the spatial cueing condition modeled here— we do not expect to observe neglect. This may explain why several groups (such as Farah et al., 1990) have failed to find object-centered neglect even though they used a paradigm similar to Driver et al. (1994).

## 4   Discussion

We have demonstrated how object-centered saccades can be performed using neurons with oculocentric receptive fields, gain modulated by the orientation of the object and the command. The same representational scheme can also account for object-centered neglect without invoking an explicit object-centered representation, i.e., representation in which neurons' receptive fields are defined in object-centered coordinates. The gain modulation by the command is consistent with the single cell data available [4], but the modulation by the orientation of the object is a prediction for future experiments.

Whether explicit object-centered representations exist, remains an empirical issue. In some cases, such representations would be computationally inefficient. In the Olson and Gettner experiment, for instance, having a stage in which motor commands are specified in object-centered coordinates does not simplify the task. Encoding the motor command in object-centered coordinates in the intermediate stage of processing requires (i) recoding the sensory input into object-centered coordinates, (ii) decoding the object-centered command into an oculocentric command, which is ultimately what the oculomotor system needs to generate the appropriate saccade. Each of these steps are computationally as complex as performing the overall transformation directly as we have done in this paper.

Therefore, gain modulation provides a simple algorithm for performing object-centered saccades. Interestingly, the same basic mechanism underlies spatial representations in other frames of reference, such as head-centered and body-centered. We have shown previously that these responses can be formalized as being basis functions of their sensory and postures inputs, a set of function which is particularly useful for sensory-motor transformations [5]. The same result applies to the SEF neurons considered in this paper, suggesting that basis functions may provide a unified theory of spatial representations in any spatial frame of reference.

## Footnotes

[1]we use the term receptive field in a general sense, meaning either receptive or motor

## References

[1] J. Driver, G. Baylis, S. Goodrich, and R. Rafal. Axis-based neglect of visual shapes. *Neuropsychologia*, 32(11):1353–1365, 1994.

[2] M. Farah, J. Brunn, A. Wong, M. Wallace, and P. Carpenter. Frames of reference for allocating attention to space: evidence from the neglect syndrome. *Neuropsychologia*, 28(4):335–47, 1990.

[3] G. Hinton. Mapping part-whole hierarchies into connectionist networks. *Artificial Intelligence*, 46(1):47–76, 1990.

[4] C. Olson and S. Gettner. Object-centered direction selectivity in the macaque supplementary eye. *Science*, 269:985–988, 1995.

[5] A. Pouget and T. Sejnowski. Spatial transformations in the parietal cortex using basis functions. *Journal of Cognitive Neuroscience*, 9(2):222–237, 1997.

[6] H. Zhou, H. Friedman, and R. von der Heydt. Edge selective cells code for figure-ground in area V2 of monkey visual cortex. In *Society For Neuroscience Abstracts*, volume 22, page 160.1, 1996.
